# Active Information Retrieval

**Tommi Jaakkola**
MIT AI Lab
Cambridge, MA
*tommi@ai.mit.edu*

**Hava Siegelmann**
MIT LIDS
Cambridge, MA
*hava@mit.edu*

## Abstract

In classical large information retrieval systems, the system responds to a user initiated query with a list of results ranked by relevance. The users may further refine their query as needed. This process may result in a lengthy correspondence without conclusion. We propose an alternative active learning approach, where the system responds to the initial user's query by successively probing the user for distinctions at multiple levels of abstraction. The system's initiated queries are optimized for speedy recovery and the user is permitted to respond with multiple selections or may reject the query. The information is in each case unambiguously incorporated by the system and the subsequent queries are adjusted to minimize the need for further exchange. The system's initiated queries are subject to resource constraints pertaining to the amount of information that can be presented to the user per iteration.

## 1 Introduction

An IR system consists of a collection of documents and an engine that retrieves documents described by users queries. In large systems, such as the Web, queries are typically too vague, and hence, an iterative process in which the users refine their queries gradually has to take place. Since much dissatisfaction of IR users stems from long, tedious repetitive search sessions, our research is targeted at shortening the search session. We propose a new search paradigm of *active information retrieval* in which the user initiates only one query, and the subsequent iterative process is led by the engine/system. The active process exploits optimum *experiment design* to permit minimal effort on the part of the user.

Our approach is related but not identical to the interactive search processes called *relevance feedback*. The primary differences pertain to the way in which the feedback is incorporated and queried from the user. In relevance feedback, the system has to deduce a set of "features" (words, phrases, etc.) that characterize the set of selected relevant documents, and use these features in formulating a new query (e.g., [5, 6]). In contrast, we cast the problem as a problem of estimation and the goal is to recover the unknown *document weights* or relevance assessments.

Our system also relates to the Scatter/Gather algorithm of browsing information systems [2], where the system initially scatters the document collection into a fixed number $k$ of clusters whose summaries are presented to the user. The user select clusters from a new sub-collection, to be scattered again into $k$ clusters, and so forth, until enumerating single documents. In our approach, documents are not discarded but rather their weighting is updated appropriately. Like many other clustering methods, the scatter/gather is based on hierarchical orderings. Overlapping clusters were recently proposed to better match real-life grouping and allow natural summarizing and viewing [4].

This short paper focuses on the underlying methodology of the active learning approach.

## 2 Active search

Let $\mathcal{X}$ be the set of documents (elements) in the database and $\mathcal{C} = \{C_1, \ldots, C_m\}$ a set of available clusters of documents for which appropriate summaries can be generated. The set of clusters typically includes individual documents and may come from a flat, hierarchical, or overlapping clustering method. The clustering need not be static, however, and could be easily defined dynamically in the search process.

Given the set of available clusters, we may choose a *query set*, a limited set of clusters that are presented to the user for selection at each iteration of the search process. The user is expected to choose the best matching cluster in this set or, alternatively, annotate the clusters with relevant/irrelevant labels (select the relevant ones). We will address both modes of operation.

The active retrieval algorithm proceeds as follows: (1) it finds a small subset $\mathcal{S}$ of clusters to present, along with their summaries, to the user; (2) waits until the user selects none, one or more of the presented clusters; (3) uses the evidence from the user's selections to update the distribution over documents or relevance assessments; (4) outputs the top documents so far, ranked by their weights, and the iteration continues until terminated by the user or the system (based on any remaining uncertainty about the relevant documents or the implied ranking).

The following sections address three primary issues: the user model, how to incorporate the information from user selections, and how to optimize the query set presented to the user. All the algorithms should scale linearly with the database size (and the size of the query set).

## 3 Contrastive selection model

We start with a contrastive selection model where the user is expected to choose only the best matching cluster in the query set. In case of multiple selections, we will interpret the marked clusters as a redefined cluster of the query set. While this interpretation will result in sub-optimal choices for the query set assuming the user consistently selects multiple clusters, the interpretation nevertheless obviates the need for modeling user's selection biases in this regard. An empty selection, on the other hand, suggests that the clusters outside the query set are deemed more likely.

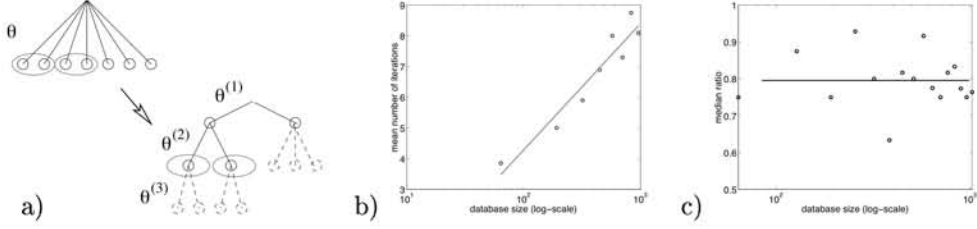

a)     b)     c)

Figure 1: a) A three level hierarchical transform of a flat Dirichlet; b) dependence of mean retrieval time on the database size (log-scale); c) median ratio of retrieval times corresponding to doubling the query set size.

To capture the ranking implied by the user selections, we define weights (distribution) $\{\theta_x\}$, $\sum_{x \in \mathcal{X}} \theta_x = 1$ over the underlying documents. We assume that the user behavior is (probabilistically) consistent with one such weighting $\theta_x^*$. The goal of a retrieval algorithm is therefore to recover this underlying weighting through interactions with the user. The resulting (approximation to) $\theta_x^*$ can be used to correctly rank the documents or, for example, to display all the documents with sufficiently large weight (cf. coverage). Naturally, $\theta_x^*$ changes from one retrieval task to another and has to be inferred separately in each task. We might estimate a user specific prior (model) over the document weights to reflect consistent biases that different users have across retrieval tasks.

We express our prior belief about the document weights in terms of a Dirichlet distribution: $P(\theta) = 1/Z \cdot \prod_{x \in \mathcal{X}} \theta_x^{\alpha_x - 1}$, where $Z = [\prod_{x \in \mathcal{X}} \Gamma(\alpha_x)]/\Gamma(\sum_{x=1}^{n} \alpha_x)$.

### 3.1 Inference

Suppose a *flat* Dirichlet distribution $P(\theta)$ over the document weights and a fixed query set $\mathcal{S} = \{C_{s_1}, \ldots, C_{s_k}\}$. We evaluate here the posterior distribution $P(\theta|y)$ given the user response $y$. The key is to transform $P(\theta)$ into a hierarchical form so as to explicate the portion of the distribution potentially affected by the user response. The hierarchy, illustrated in Figure 1a), contains three levels: selection of $\mathcal{S}$ or $\mathcal{X} \setminus \mathcal{S}$; choices within the query set $\mathcal{S}$ (of most interest to us) and those under $\mathcal{X} \setminus \mathcal{S}$; selections within the clusters $C_{s_l}$ in $\mathcal{S}$. For simplicity, the clusters are assumed to be either nested or disjoint, i.e., can be organized hierarchically.

We use $\theta_i^{(1)}, i = 1, 2$ to denote the top level parameters, $\theta_{j|1}^{(2)}$, $j = 1, \ldots, k$ for the cluster choices within the query set whereas $\theta_{x|2}^{(2)}$, $x \notin \mathcal{S}$ gives the document choices outside $\mathcal{S}$. Finally, $\theta_{x|j}^{(3)}$ for $x \in C_{s_j}$ indicate the parameters associated with the cluster $C_{s_j} \in \mathcal{S}$. The original flat Dirichlet $P(\theta)$ can be written as a product $P(\theta^{(1)})P(\theta_{\cdot|1}^{(2)})P(\theta_{\cdot|2}^{(2)}) \left[ \prod_{l=1}^{k} P(\theta_{\cdot|l}^{(3)}) \right]$ with the appropriate normalization constraints. If clusters in $\mathcal{S}$ overlap, the expansion is carried out in terms of the disjoint subsets. The parameters governing the Dirichlet component distributions are readily obtained by gathering the appropriate parameters $\alpha_x$ of the original Dirichlet (cf. [3]). For example, $\alpha_0^{(1)} = \sum_{x \in \mathcal{S}} \alpha_x$; $\alpha_{j|1}^{(2)} = \sum_{x \in C_{s_j}} \alpha_x$, for $j = 1, \ldots, k$; $\alpha_{x|2}^{(2)} = \alpha_x$, for $x \notin \mathcal{S}$; $\alpha_{x|j}^{(3)} = \alpha_x$, whenever $x \in C_{s_j}$, $j = 1, \ldots, k$.

If user selects cluster $C_{s_y}$, we will update $P(\theta_{\cdot|1}^{(2)})$ which reduces to adjusting the counts $\alpha_{y|1}^{(2)} \leftarrow \alpha_{y|1}^{(2)} + 1$. The resulting new parameters give rise to the posterior distribution $P(\theta_{\cdot|1}^{(2)}|y)$ and, by including the other components, to the overall posterior $P(\theta|y)$. If the user selects "none of these items," only the first level parameters $\theta_i^{(1)}$ will be updated.

## 3.2 Query set optimization

Our optimization criterion for choosing the query set $\mathcal{S}$ is the information that we stand to gain from querying the user with it. Let $y$ indicate the user choice, the mutual information between $y$ and the parameters $\theta$ is given by (derivation omitted)

$$I(y;\theta) \;=\; I(y;\theta_{\cdot|1}^{(2)}) = \sum_{y=1}^{k} P(y) \int P(\theta_{\cdot|1}^{(2)}|y) \log \frac{P(\theta_{\cdot|1}^{(2)}|y)}{P(\theta_{\cdot|1}^{(2)})} d\theta \tag{1}$$

$$= \sum_{y=1}^{k} P(y)\Psi(\alpha_{y|1}^{(2)} + 1) - \Psi\left(\sum_{x=1}^{k} \alpha_{x|1}^{(2)} + 1\right) + H(y) \tag{2}$$

where $P(y) = \alpha_{y|1}^{(2)}/(\sum_{x=1}^{k} \alpha_{x|1}^{(2)})$ defines our current expectation about user selection from $\mathcal{S}$; $H(y) = -\sum_{y=1}^{k} P(y) \log P(y)$ is the entropy of the selections $y$, and $\Psi(\cdot)$ is the Di-gamma function, defined as $\Psi(z) = d/dz \log \Gamma(z)$. Extending the criterion to "no selection" is trivial.

To simplify, we expand the counts $\alpha_{\cdot|1}^{(2)}$ in terms of the original (flat) counts $\alpha_x$, and define for all clusters (whether or not they appear in the query set) the weights $a_i = \sum_{x \in C_i} \alpha_x$, $b_i = a_i \Psi(a_i + 1) - a_i \log a_i$. The mutual information criterion now depends only on $a_{\mathcal{S}} = \sum_{i=1}^{k} a_{s_i} = \sum_{x \in \mathcal{S}} \alpha_x$, the overall weight of the query set and $b_{\mathcal{S}} = \sum_{i=1}^{k} b_{s_i}$ which provides an overall measure of how informative the individual clusters in $\mathcal{S}$ are. With these changes, we obtain:

$$I(y;\theta_{\cdot|1}^{(2)}) = \frac{b_{\mathcal{S}}}{a_{\mathcal{S}}} + \log(a_{\mathcal{S}}) - \Psi(a_{\mathcal{S}} + 1) \tag{3}$$

We can optimize the choice of $\mathcal{S}$ with a simple greedy method that successively finds the next best cluster index $i$ to include in the information set. This algorithm scales as $O(km)$, where $m$ is the number of clusters in our database and $k$ is the the maximal query set size in terms of the number of clusters.

Note that this simple criterion excludes nested or overlapping clusters from $\mathcal{S}$. In a more general context, the bookkeeping problem associated with the overlapping clusters is analogous to that of the Kikuchi expansion in statistical physics (cf. [7]).

## 3.3 Projection back to a flat Dirichlet

The hierarchical posterior is not a flat Dirichlet anymore. To maintain simplicity, we project it back into a flat Dirichlet in the KL-divergence sense: $P'_{\theta|y} = \arg\min_{Q_\theta} KL(P_{\theta|y}\|Q_\theta)$, where $P(\theta|y)$ is the hierarchical posterior expressed in terms of the original flat variables $\theta_x, x \in \mathcal{X}$ (but no longer a flat Dirichlet). The transformation from hierarchical to flat variables is given by: $\theta_x = \theta_1^{(1)} \theta_{j|1}^{(2)} \theta_{x|j}^{(3)}$ for

$x \in C_{s_j}$, $j = 1, \ldots, k$, and $\theta_x = \theta_2^{(1)} \theta_{x|2}^{(2)}$, for $x \in \mathcal{X} \setminus \mathcal{S}$. As a result, when $x \in C_{s_j}$ for some $j = 1, \ldots, k$ we get (derivation omitted)

$$E_{\theta|y} \log \theta_x = \Psi(\alpha_x) - \Psi\left(\sum_{z \in \mathcal{X}} \alpha_z\right) + \frac{\delta_{y,j}}{\sum_{z \in C_{s_j}} \alpha_z} - \frac{1}{\sum_{z \in \mathcal{S}} \alpha_z} \qquad (4)$$

where $y$ denotes the user selection. For $x \in \mathcal{X} \setminus \mathcal{S}$, $E_{\theta|y} \log \theta_x = \Psi(\alpha_x) - \Psi(\sum_{z \in \mathcal{X}} \alpha_z)$
If we define $r_x = E_{\theta|y} \log \theta_x$ for all $x \in \mathcal{X}$, then the counts $\beta_x$ corresponding to the flat approximation $Q_\theta$ can be found by minimizing

$$D(P_{\theta|y} \| Q_\theta) = \sum_{x \in \mathcal{X}} \left[\log \Gamma(\beta_x) - \beta_x r_x\right] - \log \Gamma\left(\sum_{x \in \mathcal{X}} \beta_x\right) + \text{const.} \qquad (5)$$

where we have omitted any terms not depending on $\beta_x$. This is a strictly convex optimization problem over the convex set $\beta_x \geq 0, x \in \mathcal{X}$ and therefore admits a unique solution. Furthermore, we can efficiently apply second order methods such as Newton-Raphson in this context due to the specific structure of the Hessian: $\mathcal{H} = D - c\mathbf{1}\mathbf{1}^T$, where $D$ is a diagonal matrix containing the derivatives of the di-gamma function[1] $\Psi'(\beta_x) = d/d\beta_x \, \Psi(\beta_x)$ and $c = \Psi'(\sum_{x \in \mathcal{X}} \beta_x)$. Each Newton-Raphson iteration requires only $\mathcal{O}(m)$ space/time.

### 3.4 Decreasing entropy

Since the query set was chosen to maximize the mutual information between the user selection and the parameters $\theta$, we get the maximal reduction in the expected entropy of the parameters: $I(y; \theta) = H(P_\theta) - E_y \, H(P_{\theta|y})$

As discussed in the previous section, we cannot maintain the true posterior but have to settle for a projection. It is therefore no longer obvious that the expected entropy of the projected posterior possesses any analogous guarantees; indeed, projections of this type typically increase the entropy. We can easily show, however, that the expected entropy is non-increasing:

$$E_y \left\{ KL(P_{\theta|y} \| P_\theta) - \min_{Q_\theta} KL(P_{\theta|y} \| Q_\theta) \right\} = H(P_\theta) - E_y \left\{ H(P'_{\theta|y}) \right\} \geq 0 \qquad (6)$$

since $P'_{\theta|y}$ is the minimizing argument. It is possible to make a stronger statement indicating that the expected entropy of the projected distribution decreases monotonically after each iteration.

**Theorem 1** *For any $\epsilon > 0$, $E_y \left\{ H(Q_{\theta|y}) \right\} \leq H(P_\theta) - \epsilon(k-1)/\alpha_\mathcal{S} + \mathcal{O}(\epsilon^2)$, where $k$ is the size of the query set and $\alpha_\mathcal{S} = \sum_{z \in \mathcal{S}} \alpha_z$.*

While this result is not tight, it does demonstrate that the projection back into a flat Dirichlet still permits a semi-linear decrease in the entropy[2]. The denominator of the first order term, i.e., $\alpha_\mathcal{S}$, can increase only by 1 at each iteration.

## 4    Annotation model

The contrastive selection approach discussed above operates a priori in a single topic mode[3]. The expectation that the user should select the best matching cluster in the query set also makes an inefficient use of the query set. We provide here an analogous development of the active learning approach under the assumption that the user classifies rather than contrasts the clusters.

The user responses are now assumed to be consistent with a noisy-OR model

$$P(y_c = 1|r^*) = 1 - (1 - q_o) \prod_{x \in c} (1 - q)^{r_x^*} \tag{7}$$

where $y_c$ is the binary relevance annotation (outcome) for a cluster $c$ in the query, $r_x^* \in \{0, 1\}$, $x \in \mathcal{X}$ are the underlying task specific relevance assignments to the elements in the database, $q$ is the probability that a relevant element in the cluster is caught by the user, and $q_o$ is the probability that a cluster is deemed "relevant" in the absence of any relevant elements. While the parameters $q_o$ and $q$ could easily be inferred from past searches, we assume here for simplicity that they are known to the search algorithm. The user annotations of different clusters in the query set are independent of each other, even for overlapping clusters.

To ensure that we can infer the unknown relevance assignments from the observables (cluster annotations), we require *identifiability*: the annotation probabilities $P(y_c = 1|r^*)$, for all $c \in \mathcal{C}$, should uniquely determine $\{r_x^*\}$. Equivalently, knowing the number of relevant documents in each cluster should enable us to recover the underlying relevance assignments. This is a property of the cluster structure and holds trivially for any clustering with access to individual elements.

The search algorithm maintains a simple independent Bernoulli model over the unknown relevance assignments: $P(r|\theta) = \prod_{x \in \mathcal{X}} \theta_x^{r_x} (1 - \theta_x)^{1-r_x}$. This gives rise to a marginal noisy-OR model over cluster annotations:

$$P(y_c = 1|\theta) = \sum_r P(y_c = 1|r) P(r|\theta) = 1 - (1 - q_o) \prod_{x \in c} (1 - \theta_x q) \tag{8}$$

The uncertainty about the relevance assignments $\{r_x\}$ makes the system beliefs about the cluster annotations *dependent* on each other. The parameters (relevance probabilities) $\{\theta_x\}$ are, of course, specific to each search task.

### 4.1    Inference and projection

Given $\hat{y}_c \in \{0, 1\}$ for a single cluster $c$, we can evaluate the posterior $P(r|\hat{y}_c, \theta)$ over the relevance assignments. Similarly to noisy-OR graphical models, this posterior can be (exponentially) costly to maintain and we instead sequentially project the posterior back into the set of independent Bernoulli distributions. The projection here is in the moments sense (*m*- projection): $P_{r;\theta'} = \arg\min_{Q_r} KL(P_{r|\hat{y}_c,\theta} \| Q_r)$, where $Q_r$ is an independent Bernoulli model. The *m*-projection preserves the posterior expectations $\theta'_{x;\hat{y}_c} = E_{r|y_c}\{r_x\}$ used for ranking the documents.

The projection yields simple element-wise updates for the parameters[4]: for $x \in c$,

$$\theta_{x;\hat{y}_c=0} = \frac{\theta_x(1-q)}{1-\theta_x q}, \quad \theta_{x;\hat{y}_c=1} = \frac{\theta_x - p_0\,\theta_{x;\hat{y}_c=0}}{1-p_0} \tag{9}$$

where $p_0 = P(y_c = 0|\theta) = (1-q_o)\prod_{x \in c}(1-\theta_x q)$ is the only parameter that depends on the cluster as a whole.

## 4.2 Query set optimization

The best single cluster $c \in \mathcal{C}$ to query has the highest mutual information between the expected user response $y_c = \{0,1\}$ and the underlying relevance assignments $r = \{r_x\}_{x \in \mathcal{X}}$, maximizing $I(y_c; r|\theta) = E_{y_c}\{KL(P_{r|\theta,y_c} \| P_{r|\theta})\}$. This mutual information cannot be evaluated in closed form, however. We use a *lower bound*:

$$I(y_c; r|\theta) \geq E_{y_c}\Big\{ \sum_{x \in c} D(\theta_{x;y_c} \| \theta_x) \Big\} \overset{def}{=} I_p(y_c; r|\theta) \tag{10}$$

where $\theta_{x;y_c}$, $x \in \mathcal{X}$ are the parameters of the m-projected posterior and $KL(\theta_{x;y_c} \| \theta_x)$ is the KL-divergence between two Bernoulli distributions with mean parameters $\theta_{x;y_c}$ and $\theta_x$, respectively.

To alleviate the concern that the lower bound would prematurely terminate the search, we note that if $I_p(r; \theta) = 0$ for all $c \in \mathcal{C}$, then $\theta_x \in \{0,1\}$ for all $x \in \mathcal{X}$. In other words, the search terminates only if we are already fully certain about the underlying relevance assignments.

The best $k$ clusters to query are those maximizing

$$I_p(y_{c_1}, y_{c_2}, \ldots, y_{c_k}; r|\theta) = E_{y_{c_1}, y_{c_2}, \ldots, y_{c_k}}\Big\{ \sum_{x \in \mathcal{X}} KL\Big(\theta_{x;y_{c_1}, y_{c_2}, \ldots, y_{c_k}} \| \theta_x\Big) \Big\} \tag{11}$$

Finding the optimal query set under this criterion (even with the m-projections) involves $\mathcal{O}(n^k 2^k)$ operations. We select the clusters sequentially while maintaining an explicit dependence on the hypothetical outcome (classification) of only the previous cluster choice. More precisely, we combine the cluster selection with conditional projections: for $k > 1$, $c_k = \arg\max_c I_p(y_c, y_{c_k}; r|\theta^{k-1})$, $\theta^k_{x;y_{c_k}} = E\{\theta^{k-1}_{x;y_{c_{k-1}}, y_{c_k}} | y_{c_k}\}$. The mutual information terms do not, however, decompose additively with the elements in the clusters. The desired $\mathcal{O}(kn)$ scaling of the selection algorithm requires a cached spline reconstruction[5].

## 4.3 Sanity check results

Figure 1b) gives the mean number of iterations of the query process as function of the database size. Each point represents an average over 20 runs with parameters

$k = 5$, $q_o = 0.05$, and $q = 0.95$. The user responses were selected on the basis of the same parameters and a randomly chosen (single) underlying element of interest. The search is terminated when the sought after element in the database has the highest rank according to $\{\theta_x\}$, $x \in \mathcal{X}$. The randomized cluster structures were relatively balanced and hierarchical. Similarly to the theoretically optimal system, the performance scales linearly with the log-database size. Results for random choice of the clusters in the query are far outside the figure.

Figure 1c), on the other hand, demonstrates that increasing the query set size appropriately reduces the interaction time. Note that since all the clusters in the query set have to be chosen prior to getting feedback from any of the clusters, doubling the query set size cannot theoretically reduce the retrieval time to a half.

## 5 Discussion

The active learning approach proposed here provides the basic methodology for optimally querying the user at multiple levels of abstraction. There are a number of extensions to the approach presented in this short paper. For example, we can encourage the user to provide confidence rated selections/annotations among the presented clusters. Both user models can be adapted to handle such selections. Analyzing the fundamental trade-offs between the size of the query set (resource constraints) and the expected completion time of the retrieval process will also be addressed in later work.

## Footnotes

[1] These derivatives can be evaluated efficiently on the basis of the highly accurate approximation to the di-gamma function.

[2] Note that the entropy of a Dirichlet distribution is not bounded from below (it is bounded from above). The manner in which the Dirichlet updates are carried out (how $\alpha_x$ change) still keeps the entropy a meaningful quantity.

[3]Dynamic redefinition of clusters partially avoids this problem.

[4]The parameters $\theta_{x;\hat{y}_{c_1},\hat{y}_{c_2},\ldots,\hat{y}_{c_k}}$ resulting from $k$ successive projections define a martingale process $E_{y_{c_1}, y_{c_2}, \ldots, y_{c_k}}\{\theta_{x;y_{c_1}, y_{c_2}, \ldots, y_{c_k}}\} = \theta_x$, $x \in \mathcal{X}$, where the expectation is taken w.r.t. to the posterior approximation.

[5]The mutual information terms for select fixed values of $p_0$ can be cached additively relative to the cluster structure. The actual $p_0$ dependence is reconstructed (quadratically) from the cached values ($I_p$ is convex in $p_0$).

## References

[1] A. C. Atkinson and A. N. Donev, Optimum experimental designs, Clarendon Press, 1992.

[2] D. R. Cutting, D. R. Karger, J. O. Pederson, J. W. Tukey, Scatter/Gather: A cluster Based Approach to Browse Document Collections, In Proceedings of the Fifteenth Annual International ACM SIGIR Conference, Denmark, June 1996.

[3] D. Heckerman, D. Geiger, and D. M. Chickering, Learning Bayesian Networks: The Combination of Knowledge and Statistical Data, Machine Learning, Vol 20, 1995.

[4] H. Lipson and H.T. Siegelmann, Geometric Neurons for Clustering, Neural Computation 12(10), August 2000

[5] J. J. Jr. Rocchio, Relevance Feedback in Information Retrieval, In The Smart System - experiments in automatic document processing, 313-323, Englewood Cliffs, NJ: Prentice Hall Inc.

[6] G. Salton and C. Buckley, Improving Retrieval Performance by Relevance Feedback, Journal of the American Society for Information Science, 41(4): 288-297, 1990.

[7] J.S. Yedidia, W.T. Freeman, Y. Weiss, Generalized Belief Propagation, Neural Information Processing Systems 13, 2001.
